# Classification on Pairwise Proximity Data

**Thore Graepel[†], Ralf Herbrich[‡],**
**Peter Bollmann-Sdorra[‡], Klaus Obermayer[†]**
Technical University of Berlin,
[‡] Statistics Research Group, Sekr. FR 6-9,
[†] Neural Information Processing Group, Sekr. FR 2-1,
Franklinstr. 28/29, 10587 Berlin, Germany

## Abstract

We investigate the problem of learning a classification task on data represented in terms of their pairwise proximities. This representation does not refer to an explicit feature representation of the data items and is thus more general than the standard approach of using Euclidean feature vectors, from which pairwise proximities can always be calculated. Our first approach is based on a combined linear embedding and classification procedure resulting in an extension of the Optimal Hyperplane algorithm to pseudo-Euclidean data. As an alternative we present another approach based on a linear threshold model in the proximity values themselves, which is optimized using Structural Risk Minimization. We show that prior knowledge about the problem can be incorporated by the choice of distance measures and examine different metrics w.r.t. their generalization. Finally, the algorithms are successfully applied to protein structure data and to data from the cat's cerebral cortex. They show better performance than K-nearest-neighbor classification.

## 1 Introduction

In most areas of pattern recognition, machine learning, and neural computation it has become common practice to represent data as feature vectors in a Euclidean vector space. This kind of representation is very convenient because the Euclidean vector space offers powerful analytical tools for data analysis not available in other representations. However, such a representation incorporates assumptions about the data that may not hold and of which the practitioner may not even be aware. And – an even more severe restriction – no domain-independent procedures for the construction of features are known [3].

A more general approach to the characterization of a set of data items is to de-

fine a proximity or distance measure between data items – not necessarily given as feature vectors – and to provide a learning algorithm with a proximity matrix of a set of training data. Since pairwise proximity measures can be defined on structured objects like graphs this procedure provides a bridge between the classical and the "structural" approaches to pattern recognition [3]. Additionally, pairwise data occur frequently in empirical sciences like psychology, psychophysics, economics, biochemistry etc., and most of the algorithms developed for this kind of data – predominantly clustering [5, 4] and multidimensional scaling [8, 6]– fall into the realm of unsupervised learning.

In contrast to nearest-neighbor classification schemes [10] we suggest algorithms which operate on the given proximity data via linear models. After a brief discussion of different kinds of proximity data in terms of possible embeddings, we suggest how the Optimal Hyperplane (OHC) algorithm for classification [2, 9] can be applied to distance data from both Euclidean and pseudo-Euclidean spaces. Subsequently, a more general model is introduced which is formulated as a linear threshold model on the proximities, and is optimized using the principle of Structural Risk Minimization [9]. We demonstrate how the choice of proximity measure influences the generalization behavior of the algorithm and apply both algorithms to real-world data from biochemistry and neuroanatomy.

## 2 The Nature of Proximity Data

When faced with proximity data in the form of a matrix $\mathbf{P} = \{p_{ij}\}$ of pairwise proximity values between data items, one idea is to embed the data in a suitable space for visualization and analysis. This is referred to as multidimensional scaling, and Torgerson [8] suggested a procedure for the linear embedding of proximity data. Interpreting the proximities as Euclidean distances in some unknown Euclidean space one can calculate an inner product matrix $\mathbf{H} = \mathbf{X}^T\mathbf{X}$ w.r.t. to the center of mass of the data from the proximities according to [8]

$$(\mathbf{H})_{ij} = -\frac{1}{2}\left(|p_{ij}|^2 - \frac{1}{\ell}\sum_{m=1}^{\ell}|p_{mj}|^2 - \frac{1}{\ell}\sum_{n=1}^{\ell}|p_{in}|^2 + \frac{1}{\ell^2}\sum_{m,n=1}^{\ell}|p_{mn}|^2\right). \quad (1)$$

Let us perform a spectral decomposition $\mathbf{H} = \mathbf{U}\mathbf{D}\mathbf{U}^T = \mathbf{X}^T\mathbf{X}$ and choose $\mathbf{D}$ and $\mathbf{U}$ such that their columns are sorted in decreasing order of magnitude of the eigenvalues $\lambda_i$ of $\mathbf{H}$. The embedding in an $n$-dimensional space is achieved by calculating the first $n$ rows of $\mathbf{X} = \mathbf{D}^{\frac{1}{2}}\mathbf{U}^T$. In order to embed a new data item characterized by a vector $\mathbf{p}$ consisting of its pairwise proximities $p_i$ w.r.t. to the previously known data items, one calculates the corresponding inner product vector $\mathbf{h}$ using (1) with $(\mathbf{H})_{ij}$, $p_{ij}$, and $p_{mj}$ replaced by $h_i$, $p_i$, and $p_m$ respectively, and then obtains the embedding $\mathbf{x} = \mathbf{D}^{-\frac{1}{2}}\mathbf{U}^T\mathbf{h}$.

The matrix $\mathbf{H}$ has negative eigenvalues if the distance data $\mathbf{P}$ were not Euclidean. Then the data can be isometrically embedded only in a pseudo-Euclidean or Minkowski space $\Re^{(n^+,n^-)}$, equipped with a bilinear form $\Phi$, which is not positive definite. In this case the distance measure takes the form $p(\mathbf{x}_i, \mathbf{x}_j) = \sqrt{\Phi(\mathbf{x}_i - \mathbf{x}_j)} = \sqrt{(\mathbf{x}_i - \mathbf{x}_j)^T\mathbf{M}(\mathbf{x}_i - \mathbf{x}_j)}$, where $\mathbf{M}$ is any $n \times n$ symmetric matrix assumed to have full rank, but not necessarily positive definite. However, we can always find a basis such that the matrix $\mathbf{M}$ assumes the form $\mathbf{M} = \text{diag}(\mathbf{I}_{n^+}, -\mathbf{I}_{n^-})$ with $n = n^+ + n^-$, where the pair $(n^+, n^-)$ is called the signature of the pseudo-Euclidean space [3]. Also in this case (1) serves to reconstruct the symmetric bilinear form, and the embedding proceeds as above with $\mathbf{D}$ replaced by $\tilde{\mathbf{D}}$, whose diagonal contains the modules of the eigenvalues of $\mathbf{H}$.

From the eigenvalue spectrum of $\mathbf{H}$ the effective dimensionality of the proximity preserving embedding can be obtained. (i) If there is only a small number of large positive eigenvalues, the data items can be reasonably embedded in a Euclidean space. (ii) If there is a small number of positive and negative eigenvalues of large absolute value, then an embedding in a pseudo-Euclidean space is possible. (iii) If the spectrum is continuous and relatively flat, then no linear embedding is possible in less than $\ell - 1$ dimensions.

## 3   Classification in Euclidean and Pseudo-Euclidean Space

Let the training set $S$ be given by an $\ell \times \ell$ matrix $\mathbf{P}$ of pairwise distances of unknown data vectors $\mathbf{x}$ in a Euclidean space, and a target class $y_i \in \{-1, +1\}$ for each data item. Assuming that the data are linearly separable, we follow the OHC algorithm [2] and set up a linear model for the classification in data space,

$$y(\mathbf{x}) = \text{sign}(\mathbf{x}^T \mathbf{w} + b). \tag{2}$$

Then we can always find a weight vector $\mathbf{w}$ and threshold $b$ such that

$$y_i(\mathbf{x}_i^T \mathbf{w} + b) \geq 1 \qquad i = 1, \ldots, \ell. \tag{3}$$

Now the optimal hyperplane with maximal margin is found by minimizing $\|\mathbf{w}\|^2$ under the constraints (3). This is equivalent to maximizing the Wolfe dual $W(\boldsymbol{\alpha})$ w.r.t. $\boldsymbol{\alpha}$,

$$W(\boldsymbol{\alpha}) = \boldsymbol{\alpha}^T \mathbf{1} - \frac{1}{2}\boldsymbol{\alpha}^T \mathbf{Y} \mathbf{X}^T \mathbf{X} \mathbf{Y} \boldsymbol{\alpha}, \tag{4}$$

with $\mathbf{Y} = \text{diag}(\mathbf{y})$, and the $\ell$-vector $\mathbf{1}$. The constraints are $\alpha_i \geq 0, \forall i$, and $\mathbf{1}^T \mathbf{Y} \boldsymbol{\alpha}^* = 0$. Since the optimal weight vector $\mathbf{w}^*$ can be expressed as a linear combination of training examples

$$\mathbf{w}^* = \mathbf{X} \mathbf{Y} \boldsymbol{\alpha}^*, \tag{5}$$

and the optimal threshold $b^*$ is obtained by evaluating $b^* = y_i - \mathbf{x}_i^T \mathbf{w}^*$ for any training example $\mathbf{x}_i$ with $\alpha_i \neq 0$, the decision function (2) can be fully evaluated using inner products between data vectors only. This formulation allows us to learn on the distance data directly.

In the Euclidean case we can apply (1) to the distance matrix $\mathbf{P}$ of the training data, obtain the inner product matrix $\mathbf{H} = \mathbf{X}^T \mathbf{X}$, and introduce it directly – without explicit embedding of the data – into the Wolfe dual (4). The same is true for the test phase, where only the inner products of the test vector with the training examples are needed.

In the case of pseudo-Euclidean distance data the inner product matrix $\mathbf{H}$ obtained from the distance matrix $\mathbf{P}$ via (1) has negative eigenvalues. This means that the corresponding data vectors can only be embedded in a pseudo-Euclidean space $\Re^{(n^+, n^-)}$ as explained in the previous section. Also $\mathbf{H}$ cannot serve as the Hessian in the quadratic programming (QP) problem (4). It turns out, however, that the indefiniteness of the bilinear form in pseudo-Euclidean spaces does not forestall linear classification [3]. A decision plane is characterized by the equation $\mathbf{x}^T \mathbf{M} \mathbf{w} = 0$, as illustrated in Fig. 1. However, Fig. 1 also shows that the same plane can just as well be described by $\mathbf{x}^T \bar{\mathbf{w}} = 0$ – as if the space were Euclidean – where $\bar{\mathbf{w}} = \mathbf{M}\mathbf{w}$ is simply the mirror image of $\mathbf{w}$ w.r.t. the axes of negative signature. For the OHC algorithm this means, that if we can reconstruct the Euclidean inner product matrix $\mathbf{X}^T \mathbf{X}$ from the distance data, we can proceed with the OHC algorithm as usual. $\bar{\mathbf{H}} = \mathbf{X}^T \mathbf{X}$ is calculated by "flipping" the axes of negative signature, i.e., with $\bar{\mathbf{D}} = \text{diag}(|\lambda_1|, \ldots, |\lambda_\ell|)$, we can calculate $\bar{\mathbf{H}}$ according to

$$\bar{\mathbf{H}} = \mathbf{U} \bar{\mathbf{D}} \mathbf{U}^T, \tag{6}$$

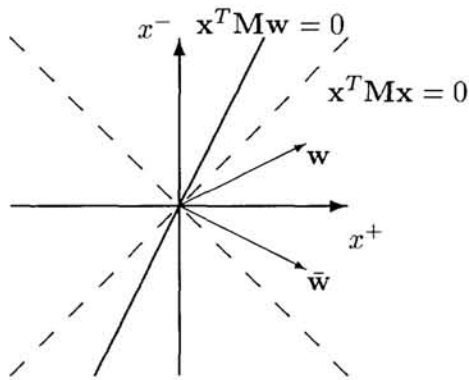

Figure 1: Plot of a decision line (thick) in a 2D pseudo-Euclidean space with signature $(1, 1)$, i.e., $\mathbf{M} = \mathrm{diag}(1, -1)$. The decision line is described by $\mathbf{x}^T \mathbf{M} \mathbf{w} = 0$. When interpreted as Euclidean it is at right angles with $\bar{\mathbf{w}}$, which is the mirror image of $\mathbf{w}$ w.r.t. the axis $x^-$ of negative signature. In physics this plot is referred to as a Minkowski space-time diagram, where $x^+$ corresponds to the space axis and $x^-$ to the time axis. The dashed diagonal lines indicate the points $\mathbf{x}^T \mathbf{M} \mathbf{x} = 0$ of zero length, the light cone.

which serves now as the Hessian matrix for normal OHC classification. Note, that $\tilde{\mathbf{H}}$ is positive semi-definite, which ensures a unique solution for the QP problem (4).

## 4 Learning a Linear Decision Function in Proximity Space

In order to cope with general proximity data (case (iii) of Section 2) let the training set $S$ be given by an $\ell \times \ell$ proximity matrix $\mathbf{P}$ whose elements $p_{ij} = p(x_i, x_j)$ are the pairwise proximity values between data items $x_i, i = 1, \ldots, \ell$, and a target class $y_i \in \{-1, +1\}$ for each data item. Let us assume that the proximity values satisfy reflexivity, $p_{ii} = 0, \forall i$, and symmetry, $p_{ij} = p_{ji}, \forall i, j$. We can make a linear model for the classification of a new data item $x$ represented by a vector of proximities $\mathbf{p} = (p_1, \ldots, p_\ell)^T$ where $p_i = p(x, x_i)$ are the proximities of $x$ w.r.t. to the items $x_i$ in the training set,

$$y(x) = \mathrm{sign}(\mathbf{p}^T \mathbf{w} + b) . \tag{7}$$

Comparing (7) to (2) we note, that this is equivalent to using the vector of proximities $\mathbf{p}$ as the feature vector $\mathbf{x}$ characterizing data item $x$. Consequently, the OHC algorithm from the previous section can be used to learn a proximity model when $\mathbf{x}$ is replaced by $\mathbf{p}$ in (2), $\mathbf{X}^T \mathbf{X}$ is replaced by $\mathbf{P}^2$ in the Wolfe dual (4), and the columns $\mathbf{p}_i$ of $\mathbf{P}$ serve as the training data.

Note that the formal correspondence does not imply that the columns of the proximity matrix are Euclidean feature vectors as used in the SV setting. We merely consider a linear threshold model on the proximities of a data item to all the training data items. Since the Hessian of the QP problem (4) is the square of the proximity matrix, it is always at least positive semi-definite, which guarantees a unique solution of the QP problem. Once the optimal coefficients $\alpha_i^*$ have been found, a test data item can be classified by determining its proximities $p_i$ from the elements $x_i$ of the training set and by using conditions (2) together with (5) for its classification.

## 5 Metric Proximities

Let us consider two examples in order to see, what learning on pairwise metric data amounts to. The first example is the minimalistic 0-1-metric, which for two objects $x_i$ and $x_j$ is defined as follows:

$$p_0(x_i, x_j) = \begin{cases} 0 & \text{if } x_i = x_j \\ 1 & \text{otherwise} \end{cases} . \tag{8}$$

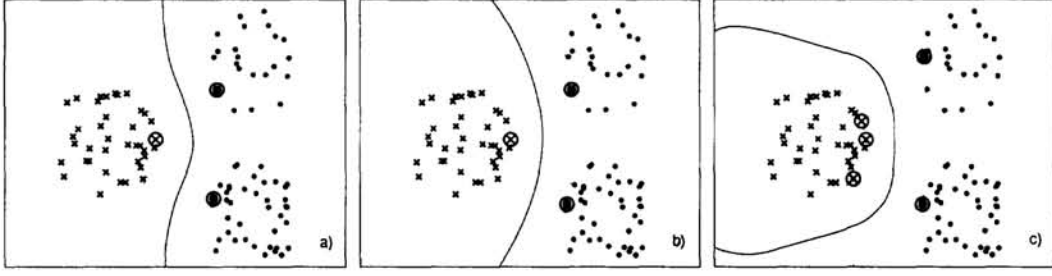

Figure 2: Decision functions in a simple two-class classification problem for different Minkowski metrics. The algorithm described in Sect. 4 was applied with **(a)** the city-block metric ($r = 1$), **(b)** the Euclidean metric ($r = 2$), and **(c)** the maximum metric ($r \to \infty$). The three metrics result in considerably different generalization behavior, and use different Support Vectors (circled).

The corresponding $\ell \times \ell$ proximity matrix $\mathbf{P}_0$ has full rank as can be seen from its non-vanishing determinant $\det(\mathbf{P}_0) = (-1)^{\ell-1}(\ell - 1)$. From the definition of the 0-1 metric it is clear that every data item $x$ not contained in the training set is represented by the same proximity vector $\mathbf{p} = \mathbf{1}$, and will be assigned to the same class. For the 0-1 metric the QP problem (4) can be solved analytically by matrix inversion, and using $\mathbf{P}_0^{-1} = (\ell - 1)^{-1}\mathbf{1}\mathbf{1}^T - \mathbf{I}$ we obtain for the classification

$$y = \text{sign}\left(\mathbf{1}^T((\ell - 1)^{-1}\mathbf{1}\mathbf{1}^T - \mathbf{I})\mathbf{Y}\mathbf{1}\right) = \text{sign}\left(\frac{1}{\ell - 1}\sum_{i=1}^{\ell} y_i\right). \qquad (9)$$

This result means, that each new data item is assigned to the majority class of the training sample, which is – given the available information – the Bayes optimal decision. This example demonstrates, how the prior information – in the case of the 0-1 metric the minimal information of identity – is encoded in the chosen distance measure.

As an easy-to-visualize example of metric distance measures on vectors $\mathbf{x} \in \Re^n$ let us consider the Minkowski $r$-metrics defined for $r \geq 1$ as

$$p(\mathbf{x}_i, \mathbf{x}_j) = \left(\sum_{\mu} |x_i^{\mu} - x_j^{\mu}|^r\right)^{1/r}. \qquad (10)$$

For $r = 2$ the Minkowski metric is equivalent to the Euclidean distance. The case $r = 1$ corresponds to the so-called city-block metric, in which the distance is given by the sum of absolute differences for each feature. On the other extreme, the maximum norm, $r \to \infty$, takes only the largest absolute difference in feature values as the distance between objects. Note that with increasing $r$ more weight is given to the larger differences in feature values, and that in the literature on multidimensional scaling [1] Minkowski metrics have been used to examine the dominance of features in human perception. Using the Minkowski metrics for classification in a toy example, we observed that different values of $r$ lead to very different generalization behavior on the same set of data points, as can be seen in Fig. 2. Since there is no apriori reason to prefer one metric over the other, using a particular metric is equivalent to incorporating prior knowledge into the solution of the problem.

| | Cat Cortex (leave-one-out) | | | | Proteins (10-fold) | | | |
|---|---|---|---|---|---|---|---|---|
| | A | V | SS | FL | H-$\alpha$ | H-$\beta$ | M | GH |
| Size of Class | 10 | 19 | 17 | 19 | 72 | 72 | 37 | 30 |
| OHC-cut-off | **3.08** | 4.62 | 6.15 | 3.08 | 0.91 | 4.01 | 0.45 | **0.00** |
| OHC-flip-axis | **3.08** | **1.54** | 4.62 | 3.08 | 0.91 | 4.01 | 0.45 | **0.00** |
| OHC-proximity | **3.08** | 4.62 | 3.08 | **1.54** | **0.45** | 3.60 | 0.45 | **0.00** |
| 1-NN | 5.82 | 6.00 | 6.09 | 6.74 | 1.65 | 3.66 | **0.00** | 2.01 |
| 2-NN | 6.09 | 4.46 | 7.91 | 5.09 | 2.01 | 5.27 | **0.00** | 3.44 |
| 3-NN | 5.29 | 2.29 | 4.18 | 4.71 | 2.14 | 6.34 | **0.00** | 2.68 |
| 4-NN | 6.45 | 5.14 | 3.68 | 5.17 | 2.46 | 5.13 | **0.00** | 4.87 |
| 5-NN | 5.55 | 2.75 | **2.72** | 5.29 | 1.65 | 5.09 | **0.00** | 4.11 |

Table 1: Classification results for Cat Cortex and Protein data. Bold numbers indicate best results.

## 6 Real-World Proximity Data

In the numerical experiments we focused on two real-world data sets, which are both given in terms of a proximity matrix **P** and class labels $y$ for each data item. The data set called "cat cortex" consists of a matrix of connection strengths between 65 cortical areas of the cat. The data was collected by Scannell [7] from text and figures of the available anatomical literature and the connections are assigned proximity values $p$ as follows: self-connection ($p = 0$), strong and dense connection ($p = 1$), intermediate connection ($p = 2$), weak connection ($p = 3$), and absent or unreported connection ($p = 4$). From functional considerations the areas can be assigned to four different regions: auditory (A), visual (V), somatosensory (SS), and frontolimbic (FL). The classification task is to discriminate between these four regions, each time one against the three others.

The second data set consists of a proximity matrix from the structural comparison of 224 protein sequences based upon the concept of evolutionary distance. The majority of these proteins can be assigned to one of four classes of globins: hemoglobin-$\alpha$ (H-$\alpha$), hemoglobin-$\beta$ (H-$\beta$), myoglobin (M), and heterogenous globins (GH). The classification task is to assign proteins to one of these classes, one against the rest.

We compared three different procedures for the described two-class classification problems, performing leave-one-out cross-validation for the "cat cortex" dataset and 10-fold cross-validation for the "protein" data set to estimate the generalization error. Table 1 shows the results. OHC-cut-off refers to the simple method of making the inner product matrix **H** positive semi-definite by neglecting projections to those eigenvectors with negative eigenvalues. OHC-flip-axis flips the axes of negative signature as described in (6) and thus preserves the information contained in those directions for classification. OHC-proximity, finally, refers to the model linear in the proximities as introduced in Section 4. It can be seen that OHC-proximity shows a better generalization than OHC-flip-axis, which in turn performs slightly better than OHC-cut-off. This is especially the case on the cat cortex data set, whose inner product matrix **H** has negative eigenvalues. For comparison, the lower part of Table 1 shows the corresponding cross-validation results for K-nearest-neighbor, which is a natural choice to use, because it only needs the pairwise proximities to determine the training data to participate in the voting. The presented algorithms OHC-flip-axis and OHC-proximity perform consistently better than K-nearest-neighbor, even when the value of $K$ is optimally chosen.

## 7   Conclusion and Future work

In this contribution we investigated the nature of proximity data and suggested ways for performing classification on them. Due to the generality of the proximity approach we expect that many other problems can be fruitfully cast into this framework. Although we focused on classification problems, regression can be considered on proximity data in an analogous way. Noting that Support Vector kernels and covariance functions for Gaussian processes are similarity measures for vector spaces, we see that this approach has recently gained a lot of popularity. However, one problem with pairwise proximities is that their number scales quadratically with the number of objects under consideration. Hence, for large scale practical applications the problems of missing data and active data selection for proximity data will be of increasing importance.

## Acknowledgments

We thank Prof. U. Kockelkorn for fruitful discussions. We also thank S. Gunn for providing his Support Vector implementation. Finally, we are indebted to M. Vingron and T. Hofmann for providing the protein data set. This project was funded by the Technical University of Berlin via the Forschungsinitiativprojekt FIP 13/41.

## References

[1] I. Borg and J. Lingoes. *Multidimensional Similarity Structure Analysis*, volume 13 of *Springer Series in Statistics*. Springer-Verlag, Berlin, Heidelberg, 1987.

[2] B. Boser, I. Guyon, and V. N. Vapnik. A training algorithm for optimal margin classifiers. In *Proceedings of the Fifth Annual Workshop on Computational Learning Theory*, pages 144–152, 1992.

[3] L. Goldfarb. *Progress in Pattern Recognition*, volume 2, chapter 9: A New Approach To Pattern Recognition, pages 241–402. Elsevier Science Publishers, 1985.

[4] T. Graepel and K. Obermayer. A stochastic self-organizing map for proximity data. Neural Computation (accepted for publication), 1998.

[5] T. Hofmann and J. Buhmann. Pairwise data clustering by deterministic annealing. *IEEE Transactions on Pattern Analysis and Machine Intelligence*, 19(1):1–14, 1997.

[6] H. Klock and J. M. Buhmann. Multidimensional scaling by deterministic annealing. In M. Pelillo and E. R. Hancock, editors, *Energy Minimization Methods in Computer Vision and Pattern Recognition*, volume 1223, pages 246–260, Berlin, Heidelberg, 1997. Springer-Verlag.

[7] J. W. Scannell, C. Blakemore, and M. P. Young. Analysis of connectivity in the cat cerebral cortex. *The Journal of Neuroscience*, 15(2):1463–1483, 1995.

[8] W. S. Torgerson. *Theory and Methods of Scaling*. Wiley, New York, 1958.

[9] V. Vapnik. *The Nature of Statistical Learning*. Springer-Verlag, Berlin, Heidelberg, Germany, 1995.

[10] D. Weinshall, D. W. Jacobs, and Y. Gdalyahu. Classification in non–metric space. In *Advances in Neural Information Processing Systems*, volume 11, 1999. in press.
